# On the Computational Utility of Consciousness

**Donald W. Mathis and Michael C. Mozer**
mathis@cs.colorado.edu, mozer@cs.colorado.edu
Department of Computer Science and Institute of Cognitive Science
University of Colorado, Boulder
Boulder, CO 80309-0430

## Abstract

We propose a computational framework for understanding and modeling human consciousness. This framework integrates many existing theoretical perspectives, yet is sufficiently concrete to allow simulation experiments. We do not attempt to explain *qualia* (subjective experience), but instead ask what differences exist within the cognitive information processing system when a person is conscious of mentally-represented information versus when that information is unconscious. The central idea we explore is that the contents of consciousness correspond to temporally persistent states in a network of computational modules. Three simulations are described illustrating that the behavior of persistent states in the models corresponds roughly to the behavior of conscious states people experience when performing similar tasks. Our simulations show that periodic settling to persistent (i.e., conscious) states improves performance by cleaning up inaccuracies and noise, forcing decisions, and helping keep the system on track toward a solution.

## 1 INTRODUCTION

We propose a computational framework for understanding and modeling consciousness. Though our ultimate goal is to explain psychological and brain imaging data with our theory, and to make testable predictions, here we simply present the framework in the context of previous experimental and theoretical work, and argue that

it is sensible from a computational perspective. We do not attempt to explain *qualia*—subjective experience and feelings. It is not clear that qualia are amenable to scientific investigation. Rather, our aim is to understand the mechanisms underlying awareness, and their role in cognition. We address three key questions:

• What are the preconditions for a mental representation to reach consciousness?

• What are the computational consequences of a representation reaching consciousness? Does a conscious state affect processing differently than an unconscious state?

• What is the computational utility of consciousness? That is, what is the computational role of the mechanism(s) underlying consciousness?

## 2  THEORETICAL FRAMEWORK

**Modular Cognitive Architecture.** We propose that the human cognitive architecture consists of a set of functionally specialized computational modules (e.g., Fodor, 1983). We imagine the modules to be organized at a somewhat coarse level and to implement processes such as visual object recognition, visual word-form recognition, auditory word and sound recognition, computation of spatial relationships, activation of semantic representations of words, sentences, and visual objects, construction of motor plans, etc. Cognitive behaviors require the coordination of many modules. For example, functional brain imaging studies indicate that there are several brain areas used for different subtasks during cognitive tasks such as word recognition (Posner & Carr, 1992).

**Modules Have Mapping And Cleanup Processes.** We propose that modules perform an associative memory function in their domain, and operate via a two-stage process: a fast, essentially feedforward input-output mapping[1] followed by a slower relaxation search (Figure 1). The computational justification for this two stage process is as follows. We assume that, in general, the output space of a module can represent a large number of states relative to the number of states that are meaningful or *well formed*—i.e., states that are interpretable by other modules or (for output modules) that correspond to sensible motor primitives. If we know which representations are well-formed, we can tolerate an inaccurate feedforward mapping, and "clean up" noise in the output by constraining it to be one of the well-formed states. This is the purpose of the relaxation step: to clean up the output of the feedforward step, resulting in a well-formed state. The cleanup process knows nothing about which output state is the best response to the input; it acts solely to enforce well-formedness. Similar architectures have been used recently to model various neuropsychological data (Hinton & Shallice, 1991; Mozer & Behrmann, 1990; Plaut & Shallice, 1993). The empirical motivation for identifying consciousness with the results of relaxation search comes from studies indicating that the contents of consciousness tend to be *coherent*, or well-formed (e.g., Baars, 1988; Crick, 1994; Damasio, 1989).

**Persistent States Enter Consciousness.** In our model, module outputs enter consciousness if they persist for a sufficiently long time. What counts as long enough

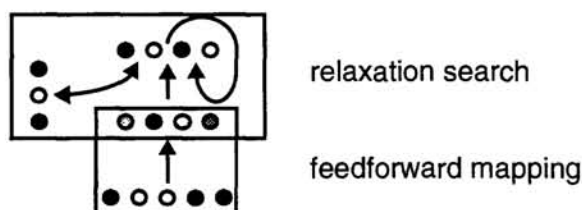

Figure 1: Modules consist of two components.

is not yet determined, but in order to model specific psychological data, we will be required to make this issue precise. At that time a specific commitment will need to be made, and this commitment must be maintained when modeling further data.

An important property of our model is that there is no hierarchy of modules with respect to awareness, in contrast to several existing theories that propose that access to some particular module (or neural processing area) is required for consciousness (e.g., Baars, 1988). Rather, information in any module reaches awareness simply by persisting long enough. The persistence hypothesis is consistent with the theoretical perspectives of Smolensky (1988), Rumelhart et al (1986), Damasio (1989), Crick and Koch (1990), and others.

## 2.1 WHEN ARE MENTAL STATES CONSCIOUS?

In our framework, the output of any module will enter consciousness if it persists in time. The persistence of an output state of a module is assured if: (1) it is a point attractor of the relaxation search (i.e., a well-formed state), and (2) the inputs to the module are relatively constant, i.e., they continue to be mapped into the same attractor basin.

While our framework appears to make strong claims about the necessary and sufficient conditions for consciousness, without an exact specification of the modules forming the cognitive architecture, it is lacking as a rigorous, testable theory. A complete theory will require not only a specification of the modules, but will also have to avoid arbitrariness in claiming that certain cognitive operations or brain regions are modules while other are not. Ultimately, one must identify the neurophysiological and neuroanatomical properties of the brain that determine the module boundaries (see Crick, 1994, for a promising step in this regard).

## 3 COMPUTATIONAL UTILITY OF CONSCIOUSNESS

For the moment, suppose that our framework provides a sensible account of awareness phenomena (demonstrating this is the goal of ongoing work.) If one accepts this, and hence the notion that a cleanup process and the resulting persistent states are required for awareness, questions about the role of cleanup in the model become quite interesting because they are equivalent to questions about the role of the mechanism underlying awareness in cognition. One question one might ask is whether there is computational utility to achieving conscious states. That is, does a system that achieves persistent states perform better than a system that does not?

Does a system that encourages settling to well-formed states perform better than a system that does not? We now show that the answer to this question is yes.

## 3.1   ADDITION SIMULATION

To examine the utility of cleanup, we trained a module to perform a simple multistep cognitive task: adding a pair of two-digit numbers in three steps.[2] We tested the system with and without cleanup and compared the generalization performance.

The network architecture (Figure 2) consists of a single module. The inputs consist of the problem statement and the current partial solution-state. The output is an updated solution-state. The module's output feeds back into its input. The problem statement is represented by four pools of units, one for each digit of each operand, where each pool uses a local encoding of digits. Partial solution states are represented by five pools, one for each of the three result digits and one for each of the two carry digits.

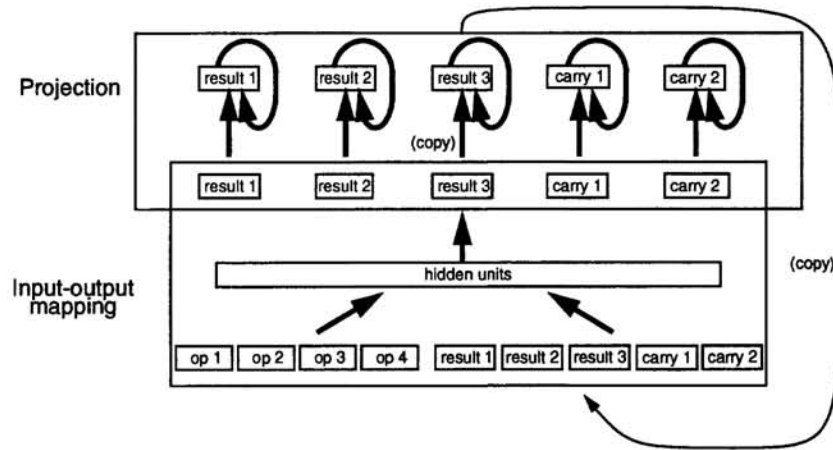

Figure 2: Network architecture for the addition task

Each addition problem was decomposed into three steps (Figure 3), each describing a transformation from one partial solution state to the next, and the mapping net was trained perform each transformation individually.

$$
\begin{array}{ccccccc}
\begin{array}{r} \text{?}\ \text{?} \\ 4\ 8 \\ +\ \ 6\ 2 \\ \hline \text{?}\ \text{?}\ \text{?} \end{array} & \text{step 1} & \begin{array}{r} \text{?}\ 1 \\ 4\ 8 \\ +\ \ 6\ 2 \\ \hline \text{?}\ \text{?}\ 0 \end{array} & \text{step 2} & \begin{array}{r} 1\ 1 \\ 4\ 8 \\ +\ \ 6\ 2 \\ \hline \text{?}\ 1\ 0 \end{array} & \text{step 3} & \begin{array}{r} 1\ 1 \\ 4\ 8 \\ +\ \ 6\ 2 \\ \hline 1\ 1\ 0 \end{array}
\end{array}
$$

Figure 3: The sequence of steps in an example addition problem

**Step 1** Given the problem statement, activate the rightmost result digit and rightmost carry digit (comprising the first partial solution).

**Step 2** Given the first partial solution, activate the next result and carry digits (second partial solution).

**Step 3** Given the second partial solution, activate the leftmost result digit (final solution).

The set of well-formed states in this domain consists of all possible combinations of digits and "don't knows" among the pools ("don't knows" are denoted by question marks in Figure 3). Local representations of digits are used within each pool, and "don't knows" are represented by the state in which no unit is active. Thus, the set of well-formed states are those in which either one or no units are active in each pool. To make these states attractors of the cleanup net, the connections were hand-wired such that each pool was a winner-take-all pool with an additional attractor at the zero state.

To run the net, a problem statement pattern is clamped on the input units, and the net is allowed to update for 200 iterations. Unit activities were updated using an incremental rule approximating continuous dynamics:

$$a_i(t) = \tau f(\sum_j w_{ij} a_j(t-1)) + (1-\tau)a_i(t-1)$$

where $a_i(t)$ is the activity of unit $i$ at time $t$, $\tau$ is a time constant in the interval [0,1], and $f(\cdot)$ is the usual sigmoid squashing function.

Figure 4 shows the average generalization performance of networks run with and without cleanup, as a function of training set size. Note that, in principle, it is not necessary for the system to have a cleanup process to learn the training set perfectly, or to generalize perfectly. Thus, it is not simply the case that no solutions exist without cleanup. The generalization results were that for any size training set, percent correct on the generalization set is always better with cleanup than without. This indicates that although the mapping network often generalizes incorrectly, the output pattern often falls within the correct attractor basin. This is especially beneficial in multistep tasks because cleanup can correct the inaccuracies introduced by the mapping network, preventing the system from gradually diverging from the desired trajectory.

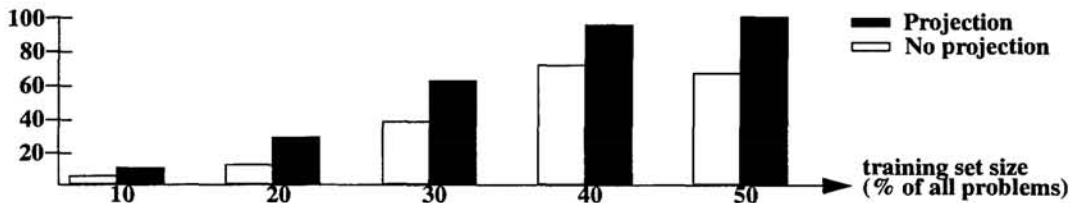

Figure 4: Cleanup improves generalization performance.

Figure 5 shows an example run of a trained network. There is one curve for each of the five result and carry pools, showing the degree of "activity" of the ultimate target pattern, **t**, for that pool as a function of time. Activity is defined to be $e^{-\|t-a\|^2}$ where **a** is the current activity pattern and **t** is the target. The network

solves the problem by passing though the correct sequence of intermediate states, each of which are temporarily persistent. This resembles the sequence of conscious states a person might experience while performing this task; each step of the problem is performed by an unconscious process, and the *results* of each of step appear in conscious awareness.

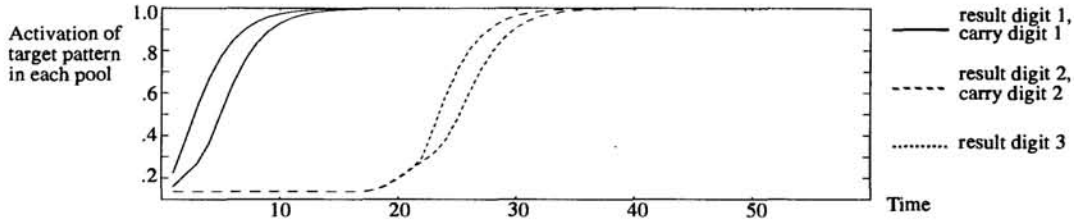

Figure 5: Network solving the addition task in three steps

## 3.2   CHOICE POINT SIMULATION

In many ordinary situations, people are required to make decisions, e.g., drive straight through an intersection or turn left, order macaroni or a sandwich for lunch. At these choice points, any of the alternative actions are reasonable a priori. Contextual information determines which action is correct, e.g., whether you are trying to drive to work or to the supermarket. Conscious decision making often occurs at these choice points, except when the task is overlearned (Mandler, 1975).

We modeled a simple form of a choice point situation. We trained a module to output sequences of states, e.g., ABCD or EFGH, where states were represented by unique activity patterns over a set of units. If the sequences shared no elements, then presenting the first element of any sequence would be sufficient to regenerate the sequence. But when sequences overlap, choice points are created. For example, with the sequences ABCD and AEFG, state A can be followed by either B or E.

We show that cleanup allows the module to make a decision and complete one of the two sequences. Figure 6 shows the operation of the module with and without cleanup following presentation of an A after training on the sequence pair ABCD and AEFG. There is one curve for each state, showing the activation of that state (defined as before), as a function of time. When the network is run with cleanup, although both states B and E are initially partially activated, the cleanup process maps this ill-formed state to state B, and the network then correctly completes the sequence ABCD. Without cleanup, the initial activation of states B and E causes a blending of the two sequences ABCD and AEFG and the state degenerates.[3]

Although the arithmetic and choice point tasks seem simple in part because we predefined the set of well-formed states. However, because the architecture segre-

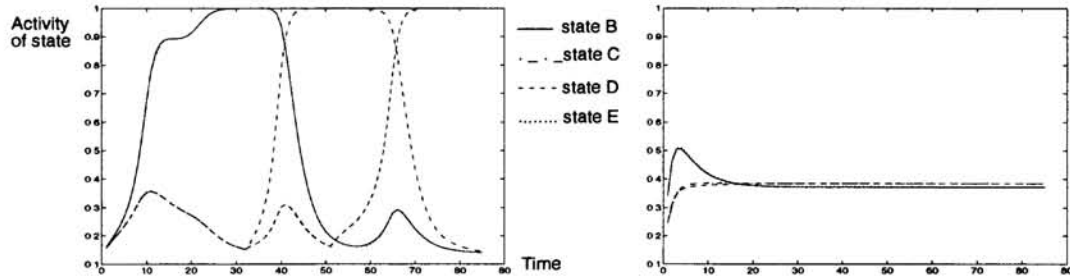

Figure 6: Decision-point task with and without cleanup

gates knowledge of well-formedness from knowledge of how to solve problems in the domain, well-formedness could be learned simultaneously with, or prior to learning the task. One could imagine training the cleanup network to autoassociate states it observes in the domain before or during training using an unsupervised or self-supervised procedure.

# 4  COMPUTATIONAL CONSEQUENCES OF PERSISTENT STATES

In a network of modules, a persistent well-formed state in one module exerts larger influences on the state of other modules than do transient or ill-formed states. As a result the dynamics of the system tends to be dominated by well-formed persistent states. We show this in a final simulation.

The network consisted of two modules, A and B, connected in a simple feedforward cascade. Each module's cleanup net was trained to have ten attractors, locally represented in a winner-take-all pool of units. The mapping network of module B was trained to map the attractors of module A to attractors in B in a one-to-one fashion. Thus, state $\alpha_1$ in module A is mapped to $\beta_1$ in module A, $\alpha_2$ to $\beta_2$, etc.

Module B was initialized to a well-formed state $\beta_1$, and the output state of module A was varied, creating three conditions. In the *persistent well-formed* condition, module A was clamped in the well-formed state $\alpha_2$ for 50 time steps. In the *transient well-formed* condition, module A was clamped in state $\alpha_2$ for only 30 time steps. And in the *ill-formed* condition, module A was clamped in an ill-formed state in which two states, $\alpha_2$ and $\alpha_3$, were both partially active. Figure 7 shows the subsequent activation of state $\beta_2$ in module B as a function of time. Module B undergoes a transition from state $\beta_1$ to state $\beta_2$ only in the persistent well-formed condition. This indicates that the *conjunction* of well-formedness and persistence is required to effect a transition from one state to another.

# 5  CONCLUSIONS

Our computational framework and simulation results suggest the following answers to our three key questions:

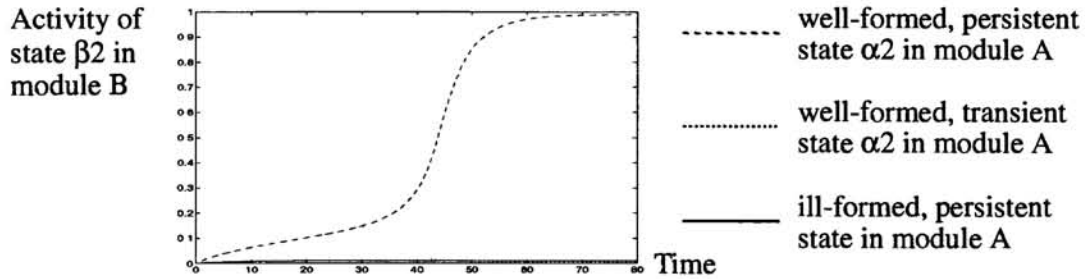

Figure 7: Well-formedness and persistence are both required for attractor transitions.

- In order to reach consciousness, the output of a module must be both persistent and semantically well-formed, and must not initiate an overlearned process.

- The computational consequences of conscious (persistent) representations include exerting larger influences on the cognitive system, resulting in increased ability to drive response processes such as verbal report.

- The computational utility of consciousness in our model lies in the ability of cleanup to "focus" cognition, by keeping the system close to states which are semantically meaningful. Because the system has learned to process such states, performance is improved.

## Footnotes

[1]We do not propose that this process is feedforward at the neural level. Rather, we mean that any iterative refinement of the output over time is unimportant and irrelevant.

[2]Of course, we don't believe that there is a brain module dedicated to addition problems. This choice was made because addition is an intuitive example of a multistep task.

[3]In this simulation, we are not modeling the role of context in helping to select one sequence or another; we are simply assuming that either sequence is valid in the current context. The nature of the model does not change when we consider context. Assuming that the domain is not highly overlearned, the context will not strongly evoke one alternative action or the other in the feedforward mapping, leading to partial activation of multiple states, and the cleanup process will be needed to force a decision.

## References

Baars, B. J. (1988) *A Cognitive Theory of Consciousness*, Cambridge University Press.

Crick, F. (1994) The astonishing hypothesis: The scientific search for the soul. Scribner.

Crick, F., & Koch, C. (1990) Towards a neurobiological theory of consciousness. *Sem. Neuro.*, 2: 263-275

Damasio, A. (1989) The brain binds entities and events by multiregional activation from convergence zones. *Neural Computation*, 1, 123-132

Fodor, J. A. (1983) *The modularity of mind: An essay on faculty psychology.* Cambridge, MA: MIT Press.

Hinton, G. E., & Shallice, T. (1991) Lesioning an attractor network: Investigations of acquired dyslexia., *Psych. Rev.*, 98: 74-95

Mandler, G. (1975) Consciousness: Respectable, useful and probably necessary. In Information Processing and Cognition, The Loyola Symposium, R. Solso (Ed.). Erlbaum.

Mozer, M. C., & Behrmann, M. (1990). On the interaction of spatial attention and lexical knowledge: A connectionist account of neglect dyslexia. *Cognitive Neuroscience*, 2, 96-123.

Plaut, D. C., & Shallice, T. (1993) Perseverative and semantic influences on visual object naming errors in optic aphasia: A connectionist account. *J. Cog. Neuro.*, 5(1): 89-117

Posner, M. I., & Carr, T. (1992) Lexical access and the brain: Anatomical constraints on cognitive models of word recognition. *American Journal of Psychology*, 105(1): 1-26

Rumelhart, D. E., Smolensky, P., McClelland, J. L., & Hinton, G. E. (1986) Schemata and sequential thought in PDP models. In J. L. McClelland & D. E. Rumelhart (Eds.), *Parallel Distributed Processing*, Vol. 2. Cambridge, MA: MIT Press.

Smolensky, P. (1988) On the proper treatment of connectionism. *Brain Behav. Sci.*, 11: 1-74